# Kernels on Structured Objects Through Nested Histograms

**Marco Cuturi**
Institute of Statistical Mathematics
Minami-azabu 4-6-7, Minato ku,
Tokyo, Japan.

**Kenji Fukumizu**
Institute of Statistical Mathematics
Minami-azabu 4-6-7, Minato ku,
Tokyo, Japan.

## Abstract

We propose a family of kernels for structured objects which is based on the bag-of-components paradigm. However, rather than decomposing each complex object into the single histogram of its components, we use for each object a family of nested histograms, where each histogram in this hierarchy describes the object seen from an increasingly granular perspective. We use this hierarchy of histograms to define elementary kernels which can detect coarse and fine similarities between the objects. We compute through an efficient averaging trick a mixture of such specific kernels, to propose a final kernel value which weights efficiently local and global matches. We propose experimental results on an image retrieval experiment which show that this mixture is an effective template procedure to be used with kernels on histograms.

## 1 Introduction

Kernel methods have shown to be competitive with other techniques in classification or regression tasks where the input data lie in a vector space. Arguably, this success rests on two factors: first, the good ability of kernel algorithms, such as the support vector machine, to generalize and provide a sparse formulation for the underlying learning problem; second, the capacity of nonlinear kernels, such as the polynomial and gaussian kernels, to quantify meaningful similarities between vectors, notably non-linear correlations between their components. Using kernel machines with non-vectorial data (e.g., in bioinformatics, image and text analysis or signal processing) requires more arbitrary choices, both to represent the objects in a malleable form, and to choose suitable kernels on these representations. The challenge of using kernel methods on real-world data has thus recently fostered many proposals for kernels on complex objects, notably strings, trees, images or graphs to cite a few.

In common practice, most of these objects can be regarded as structured aggregates of smaller components, and the coarsest approach to study such aggregates is to consider them directly as bags of components. In the field of kernel methods, such a representation has not only been widely adopted (Haussler, 1999; Joachims, 2002; Schölkopf et al., 2004), but it has also spurred the proposal of kernels better suited to the geometry of the underlying histograms (Kondor & Jebara, 2003; Lafferty & Lebanon, 2005; Hein & Bousquet, 2005; Cuturi et al., 2005). However, one of the drawbacks of the bag-of-components representation is that it implicitly assumes that each component sampled in the object has been generated independently from an identical distribution. While this viewpoint may translate into adequate properties for some learning tasks, such as translation or rotation invariance when using histograms of colors to manipulate images (Chapelle et al., 1999), it may however appear too restrictive when such a strong invariance may just be too coarse to be of practical use.

A possible way to cope with this limitation is to expand artificially the size of the components' space, either by considering families of larger components to take into account more contextual information, or by considering histograms which index both components and their possible location in the object (Rätsch & Sonnenburg, 2004). As one would expect, these histograms are usually sparse and need to be regularized using ad-hoc rules and prior knowledge (Leslie et al., 2003) before being directly compared using kernels on histograms. For sequential data, other state-of-the-art methods compute an optimal alignment between the sequences based on elementary operations such as substitutions, deletions and insertions of components. Such alignment scores may yield positive definite (p.d.) kernels if particular care is taken to adapt them (Vert et al., 2004) and have shown very competitive performances. However, their computational cost can be prohibitive when dealing with large datasets, and can only be applied to sequential data. Following these contributions, we propose

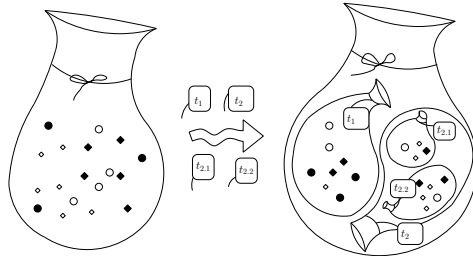

Figure 1: From the bag of components representation to a set of nested bags, using a set of labels.

in this paper new families of kernels which can be easily tuned to detect both coarse and fine similarities between the objects, in a range spanned from kernels which only consider coarse histograms to kernels which only detect strict local matches. To size such types of similarities between two objects, we elaborate on the elementary bag-of-components perspective to consider instead families of nested histograms (indexed by a set of hierarchical labels to be defined) to describe each object. In this framework, the root label corresponds to the global representation introduced before, while longer labels represent a specific condition under which the components have been sampled. We then define kernels that take into account mixtures of similarities, spanning from detailed resolutions which only compare the smallest bags to the coarsest one. This trade-off between fine and coarse perspectives sets an averaging framework to define kernels, which we introduce formally in Section 2. This theoretical framework would not be tractable without an efficient factorization detailed in Section 3 which yields computations which grow linearly in time and space with respect to the number of labels to evaluate the value of the kernel. We then provide experimental results in Section 4 on an image retrieval task which shows that the methodology improves the performance of kernel based state-of-the art techniques in this field with a low extra computational cost.

## 2   Kernels Defined through Hierarchies of Histograms

In the kernel literature, structured objects are usually represented as histograms of components, e.g., images as histograms of colors and/or features, texts as bags of words and sequences as histograms of letters or $n$-grams. The obvious drawback of this representation is that it usually loses all the contextual information which may be useful to characterize each sampled component in the original object. One may instead create families of histograms, indexed by specific sampling conditions:

- In image analysis, create color or feature histograms following a prior partition of the image into predefined patches, as in (Grauman & Darrell, 2005). Another possibility would be to define families of histograms, all for the same image, which would consider increasingly granular discretizations of the color space.

- In sequence analysis, extract local histograms which may correspond to predefined regions of the original sequence, as in (Matsuda et al., 2005). A different option would be to associate to each histogram a context of arbitrary length, e.g. by considering the 26 histogram of letters sampled just after the letters $\{A, B, \cdots, Z\}$, or the $26 \times 26$ histograms of letters after contexts $\{AA, AB, \cdots, ZZ\}$.

- In text analysis, use histograms of words found after grammatical categories of increasing complexity, such as verbs, nouns, articles or adverbs.
- For synchronous time series (e.g. financial time series or gene expression profiles), define a reference series (e.g. an index or a specific gene) and decompose each of the subsequent series into histograms of values conditioned to the value of the reference series.

We write $\mathcal{L}$ for an arbitrary index set to label such specific histograms. Structured objects are thus represented as a family $\mu$ of $M_{\mathcal{L}}(\mathcal{X}) \overset{\text{def}}{=} (M_+^b(\mathcal{X}))^{\mathcal{L}}$, that is $\mu = \{\mu_t\}_{t \in \mathcal{L}}$ where for each $t \in \mathcal{L}$, $\mu_t$ is a bounded measure of $M_+^b(\mathcal{X})$. We write $|\mu|$ for $\sum_{t \in \mathcal{L}} |\mu_t|$.

## 2.1 Local Similarities Between Measures

To compare two objects under the light of any sampling condition $t$, that is comparing their respective decompositions as measures $\mu_t$ and $\mu_t'$, we make use of an arbitrary p.d. kernel $k$ on $M_+^b(\mathcal{X})$ to which we will refer as the base kernel throughout the paper. For interpretation purposes only, we will assume in the following sections that $k$ is an infinitely divisible kernel which can be written as $k = e^{-\frac{1}{\lambda}\psi}, \quad \lambda > 0$, where $\psi$ is a negative definite (Berg et al., 1984) kernel on $M_+^b(\mathcal{X})$, or equivalently $-\psi$ is a conditionally p.d. kernel. Note also that $k$ has to be p.d. not only on probability measures, but on any bounded measure. For two elements $\mu, \mu'$ of $M_{\mathcal{L}}(\mathcal{X})$ and a given element $t \in \mathcal{L}$, the kernel

$$k_t(\mu, \mu') \overset{\text{def}}{=} k(\mu_t, \mu_t')$$

quantifies the similarity of $\mu$ and $\mu'$ by measuring how similarly their components were observed with respect to label $t$. For two different labels $s$ and $t$ of $\mathcal{L}$, $k_s$ and $k_t$ can be associated through polynomial combinations with positive coefficients to result in new kernels, notably their sum $k_s + k_t$ or their product $k_s k_t$. This is particularly adequate if some complementarity is assumed between $s$ and $t$, so that their combination can provide new insights for a given learning task. If on the contrary these labels are assumed to be similar, then they can be regarded as a grouped label $\{s\} \cup \{t\}$ and result in the kernel

$$k_{\{s\} \cup \{t\}}(\mu, \mu') \overset{\text{def}}{=} k(\mu_s + \mu_t, \mu_s' + \mu_t'),$$

which will measure the similarity of $m$ and $m'$ under *both* $s$ or $t$ labels. Let us give an intuition for this definition by considering two texts $A, B$ built up with words from a dictionary $\mathcal{D}$. As an alternative to the general histograms of words $\theta^A$ and $\theta^B$ of $M_+^b(\mathcal{D})$, one may consider for instance $\theta_{\text{can}}^A, \theta_{\text{may}}^A$ and $\theta_{\text{can}}^B, \theta_{\text{may}}^B$, the respective histograms of words that follow the words `can` and `may` in texts A and B respectively. If one considers that `can` and `may` are different words, then the following kernel quantifies the similarity of $A$ and $B$ taking advantage of this difference:

$$k_{\{\text{can}\}, \{\text{may}\}}(A, B) = k(\theta_{\text{can}}^A, \theta_{\text{can}}^B) \times k(\theta_{\text{may}}^A, \theta_{\text{may}}^B).$$

If on the contrary one decides that `can` and `may` are equivalent, an adequate kernel would first merge the histograms, and then compare them:

$$k_{\{\text{can,may}\}}(A, B) = k(\theta_{\text{can}}^A + \theta_{\text{may}}^A, \theta_{\text{can}}^B + \theta_{\text{may}}^B).$$

The previous formula can be naturally extended to define kernels indexed on a set $T \subset \mathcal{L}$ of grouped labels, through

$$k_T(\mu, \mu') \overset{\text{def}}{=} k(\mu_T, \mu_T'), \text{ where } \mu_T \overset{\text{def}}{=} \sum_{t \in T} \mu_t \text{ and } \mu_T' \overset{\text{def}}{=} \sum_{t \in T} \mu_t'.$$

## 2.2 Resolution Specific Kernels

Having defined a family of kernels $\{k_T, T \subset \mathcal{L}\}$ which can detect conditional similarities between two elements of $M_{\mathcal{L}}(\mathcal{X})$ given a subset $T$ of $\mathcal{L}$, we define in this section different ways to combine them to obtain a kernel which can take into account all of their histograms. Let $P$ be a finite partition of $\mathcal{L}$, that is a finite family $P = (T_1, ..., T_n)$ of sets of $\mathcal{L}$, such that $T_i \cap T_j = \varnothing$ if $1 \le i < j \le n$ and $\bigcup_{i=1}^n T_i = \mathcal{L}$. We write $\mathcal{P}(\mathcal{L})$ for the set of all partitions of $\mathcal{L}$. Consider now the kernel defined by a partition $P$ as

$$k_P(\mu, \mu') \overset{\text{def}}{=} \prod_{i=1}^n k_{T_i}(\mu, \mu'). \tag{1}$$

The kernel $k_P$ quantifies the similarity between two objects by detecting their joint similarity under all possible labels of $\mathcal{L}$, assuming *a priori* that certain labels can be grouped together, following the subsets $T_i$ enumerated in the partition $P$. Note that there is some arbitrary in this definition since a simple multiplication of base kernels $k_{T_i}$ is used to define $k_P$, rather than any other polynomial combination. We follow in that sense the convolution kernels (Haussler, 1999) approach, and indeed, for each partition $P$, $k_P$ can be regarded as a convolution kernel. More precisely, the multiplicative structure of Equation (1) quantifies how similar two objects are given a partition $P$, in a way that imposes for the objects to be similar according to all subsets $T_i$. If the base kernel $k$ can be written as $k = e^{-\frac{1}{\lambda}\psi}$, where $\psi$ is a negative definite kernel, then $k_P$ can be expressed as the exponential of minus

$$\psi_P(\mu, \mu') \stackrel{\text{def}}{=} \sum_{i=1}^{n} \psi_{T_i}(\mu, \mu') = \sum_{i=1}^{n} \psi(\mu_{T_i}, \mu'_{T_i}),$$

a quantity which penalizes local differences between the decompositions of $\mu$ and $\mu'$ over $\mathcal{L}$, as opposed to the coarsest approach where $P = \{\mathcal{L}\}$ and only $\psi(\sum_t \mu_t, \sum_t \mu'_t)$ is considered.

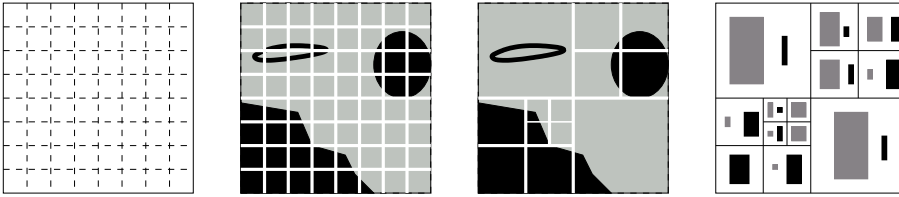

Figure 2: A useful set of labels $\mathcal{L}$ for images which would focus on pixel localization can be represented by a grid, such as the $8 \times 8$ one represented above. In this case $P_3$ corresponds to the $4^3$ windows presented in the left image, $P_2$ to the 16 larger squares obtained when grouping 4 small windows, $P_1$ to the image divided into 4 equal parts and $P_0$ is simply the whole image. Any partition $P$ of the image which complies with the hierarchy $P_0^3$ in the example above, can in turn be used to represent an image as a family of sub-probability measures, which reduces in the case of two-color images to binary histograms as illustrated in the right-most image. For two images, these respective histograms can be directly compared through the kernel $k_P$.

As illustrated in Figure 2, where images are summarized through histograms indexed by patches, a partition of $\mathcal{L}$ reflects a given belief on how patches may or may not be associated or split to focus on local dissimilarities. Hence, all partitions contained in the set $\mathcal{P}(\mathcal{L})$ of all possible partitions[1] are not likely to be equally meaningful given that some labels may a natural form of grouping. If the index is built to highlight differences in locations, one would naturally favor mergers between neighboring indexes. If one uses a Markovian analysis, that is consider histograms of components conditioned by contexts, a natural way to group contexts would be to group them according to their semantic or grammatical content for text analysis or according to their suffix for sequence analysis.

Such meaningful partitions can be intuitively obtained when a hierarchical structure which groups elements of $\mathcal{L}$ together is known a priori. A hierarchy on $\mathcal{L}$, such as the triadic hierarchy shown in Figure 3, is a family

$$(P_d)_{d=0}^D = \{P_0 = \{\mathcal{L}\}, .., P_D = \{\{t\}, t \in \mathcal{L}\}\}$$

of partitions of $\mathcal{L}$. To provide a hierarchical information, the family $(P_d)_{d=0}^D$ is such that any subset present in a partition $P_d$ is strictly included in a (unique by definition of a partition) subset from the coarser partition $P_{d-1}$. This is equivalent to stating that each subset $T$ in a partition $P_d$ is divided in $P_{d+1}$ as a partition of $T$ which is not $T$ itself. We write $s(T)$ for this partition (e.g., in Figure 3, $s(1) = \{1_1, \cdots, 1_9\}$) and name its elements the siblings of $T$. Consider now the subset $\mathcal{P}_D \subset \mathcal{P}(\mathcal{L})$ of all partitions of $\mathcal{L}$ obtained by using only sets contained in the collection $P_0^D \stackrel{\text{def}}{=} \bigcup_{d=0}^D P_d$, namely $\mathcal{P}_D \stackrel{\text{def}}{=} \{P \in \mathcal{P}(\mathcal{L}) \text{ s.t. } \forall T \in P, T \in P_0^D\}$. The set $\mathcal{P}_D$ contains both the coarsest and the finest resolutions, respectively $P_0$ and $P_D$, but also all variable resolutions for sets enumerated in $P_0^D$, as can be seen for instance in the third image of Figure 2.

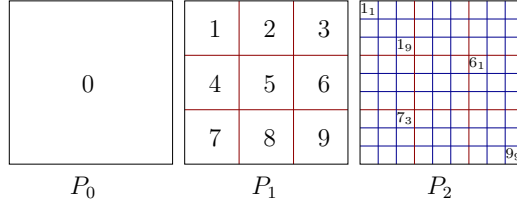

Figure 3: A hierarchy generated by two successive triadic partitions.

## 2.3 Averaging Resolution Specific Kernels

Each partition $P$ contained in $\mathcal{P}_D$ provides a resolution to compare two objects, which generates a large family of kernels $k_P$ when $P$ spans $\mathcal{P}_D$. Some partitions are likely to be better suited for certain tasks, which may call for an efficient estimation scheme to select an optimal partition for a given task. This would be similar in spirit to estimating a maximum a posteriori model for the data and use it consequently to compare the objects. We take in this section a different direction which has a more Bayesian flavor by considering an averaging of such kernels based on a prior on the set of partitions. In practice, this averaging favours objects which share similarities under a large collection of resolutions, and may also be interpreted as a Bayesian averaging of convolution kernels (Haussler, 1999).

**Definition 1** *Let $\mathcal{L}$ be an index set endowed with a hierarchy $(P_d)_{d=0}^{D}$, $\pi$ be a prior measure on the corresponding set of partitions $\mathcal{P}_D$ and $k$ a base kernel on $M_+^b(\mathcal{X}) \times M_+^b(\mathcal{X})$. The averaged kernel $k_\pi$ on $M_{\mathcal{L}}(\mathcal{X}) \times M_{\mathcal{L}}(\mathcal{X})$ is defined as*

$$k_\pi(\mu, \mu') = \sum_{P \in \mathcal{P}_D} \pi(P) \, k_P(\mu, \mu'). \tag{2}$$

As can be observed in Equation (2), the kernel automatically detects in the range of all partitions the ones which provide a good match between the compared objects, to increase subsequently the resulting similarity score. Also note that in an image-analysis context, the pyramid-matching kernel proposed in (Grauman & Darrell, 2005) only considers the original partitions of the hierarchy $(P_d)_{d=0}^{D}$, while Equation (2) considers all possible partitions of $\mathcal{P}_D$. This can be carried out with little cost if an adequate set of priors $\pi$ is selected as seen below.

# 3 Kernel Computation

We provide in this section hierarchies $(P_d)_{d=0}^{D}$ and priors $\pi$ for which the computation of $k_\pi$ is both meaningful and tractable, yielding namely a computational time to calculate $k_\pi$ which is loosely upperbounded by $D \times \operatorname{card} \mathcal{L} \times c(k)$ where $c(k)$ is the time required to compute the base kernel.

## 3.1 Partitions Generated by Branching Processes

All partitions $P$ of $\mathcal{P}_D$ can be generated through the following rule, starting from the initial root partition $P := P_0 = \{\mathcal{L}\}$. For each set $T$ of $P$:

1. either leave the set as it is in $P$ with probability $1 - \varepsilon_T$,
2. either replace it by its siblings in $s(T)$ with probability $\varepsilon_T$, and reapply this rule to each sibling unless they belong to the finest partition $P_D$.

The resulting prior for $\mathcal{P}_D$ depends on the overall coarseness of the considered partitions, and can be tuned through parameters $\varepsilon_T$ to favor adaptively coarse or fine partitions. For a partition $P \in \mathcal{P}_D$, $\pi(P) = \prod_{T \in P}(1 - \varepsilon_T) \prod_{T \in \overset{\circ}{P}}(\varepsilon_T)$, where the set $\overset{\circ}{P} = \{T \in P_0^D \text{ s.t. } \exists V \in P, V \subsetneq T\}$ gathers all coarser sets belonging to coarser resolutions than $P$, and can be regarded as the set of all ancestors in $P_0^D$ of sets enumerated in $P$.

## 3.2 Factorization of $k_\pi$

We use the branching-process prior can be used to factorize the formula in Equation (2):

**Proposition 2** *For two elements $\mu, \mu'$ of $M_\mathcal{L}(\mathcal{X})$, define for $T$ spanning recursively all sets contained in $P_D, P_{D-1}, ..., P_0$ the quantity $K_T$ below; then $k_\pi(\mu, \mu') = K_\mathcal{L}$.*

$$K_T = (1 - \varepsilon_T)k_T(\mu, \mu') + \varepsilon_T \prod_{U \in s(T)} K_U.$$

**Proof**

The proof follows from a factorization which uses the branching process prior used for the tree generation, and can be derived from the proof of (Catoni, 2004, Proposition 5.2). The opposite figure underlines the importance of incorporating to each node $K_T$ a weighted product of the sibling kernel evaluations $K_U$. The update rule for the computation of $k_\pi$ takes into account the branching process prior by weighting the kernel $k_T$ with all values $k_{t_i}$ obtained for finer resolutions $t_i$ in $s(T)$.
∎

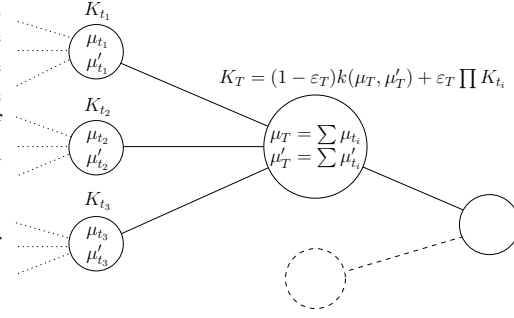

If the hierarchy of $\mathcal{L}$ is such that the cardinality of $s(T)$ is fixed to a constant $\alpha$ for any set $T$, typically $\alpha = 4$ for images in the case described in Figure 2, then the computation of $k_\pi$ is upperbounded by $(\alpha^{D+1} - 1)c(k)$. This complexity is also upperbounded by the total amount of components considered in the compared objects, as in (Cuturi & Vert, 2005) for instance.

## 3.3 Choosing the Base Kernel

Any kernel on $M_+^b(\mathcal{X})$ can be used to comply with the terms of Definition 1 and apply an average scheme on families of measures. We also note that an even more general formulation can be obtained by using a different kernel $k_t$ for each label $t$ of $\mathcal{L}$, without altering the overall applicability of the factorization above. However, we only consider in this discussion a unique choice $k$ for all $t \in \mathcal{L}$.

First, one can note that kernels such as the information diffusion kernel (Lafferty & Lebanon, 2005) and variance based kernels (Kondor & Jebara, 2003; Cuturi et al., 2005) may not work in this setting since they are not p.d., nor sometimes defined, on the whole of $M_+^b(\mathcal{X})$. The most adequate geometry of $M_+^b(\mathcal{X})$, following the denormalization scheme proposed in (Amari & Nagaoka, 2001, p.47), may arguably be derived from the Riemannian embedding $\nu \mapsto \sqrt{\nu}$, where the Euclidian distance between two measures in this representation is equal to the geodesic distance between $\nu$ and $\nu'$ in $M_+^b(\mathcal{X})$ endowed with the Fisher metric, as expressed in $\psi_{H_2}$ below. More generally, one can consider the whole family of kernels for bounded measures described in (Hein & Bousquet, 2005) to choose the base kernel $k$, namely the family of Hilbertian metrics $\psi$ such that $k = e^{-\frac{1}{\lambda}\psi}$. We thus use in our experiments the Jensen divergence, the $\chi^2$ distance, the total variation, and two variations of the Hellinger distance:

$$\psi_{JD}(\theta, \theta') = h\left(\frac{\theta + \theta'}{2}\right) - \frac{h(\theta) + h(\theta')}{2}, \quad \psi_{\chi^2}(\theta, \theta') = \sum_i \frac{(\theta_i - \theta_i')^2}{\theta_i + \theta_i'},$$

$$\psi_{TV}(\theta, \theta') = \sum_i |\theta_i - \theta_i'|, \quad \psi_{H_2}(\theta, \theta') = \sum_i |\sqrt{\theta_i} - \sqrt{\theta_i'}|^2, \quad \psi_{H_1}(\theta, \theta') = \sum_i |\sqrt{\theta_i} - \sqrt{\theta_i'}|.$$

# 4 Experiments in Image Retrieval

We present in this section experiments inspired by the image retrieval task first considered in (Chapelle et al., 1999) and reused in (Hein & Bousquet, 2005). Our dataset was also extracted from the Corel Stock database and includes 12 families of labeled images, each class containing

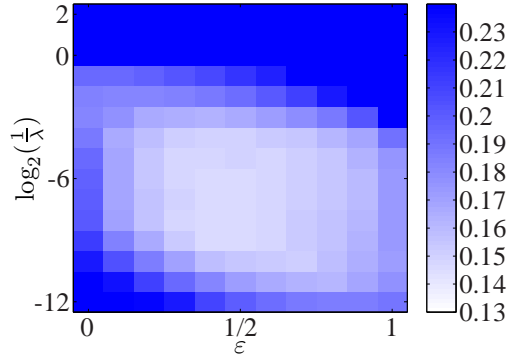

Figure 4: Misclassification rate on the corel experiment, using the Hellinger $H_1$ distance between histograms coupled with one-vs-all SVM classification ($C = 100$) as a function of $\lambda$ and $\varepsilon$. $\frac{1}{\lambda}$ is taken in $\{2^{-12}, \cdots, 2^2\}$ while $\varepsilon$ spans $\{0, 0.1, \cdots, 0.9, 1\}$. $\varepsilon$ controls the granularity of the averaging kernel, ranging from the coarsest perspective ($\varepsilon = 0$) when only the global histogram is used, to the finest one ($\varepsilon = 1$) when only the finest histograms are considered. Dark values represent error rates which are *greater* or equal to 24%. The central values are roughly 14.5% while the best value obtained in the columns $\varepsilon = 0$ and $\varepsilon = 1$ are 18.4% and 17.3% respectively

100 color images of $256 \times 384$ pixels. The families depict images of *bears, African specialty animals, monkeys, cougars, fireworks, mountains, office interiors, bonsais, sunsets, clouds, apes* and *rocks and gems*. The database is randomly split into balanced sets of 800 training images and 400 test images. The task consists in classifying the test images with the rule learned by training 12 one-versus-all SVM's on the learning fold. Note that previous work conducted in (Chapelle et al., 1999) illustrates the competitiveness of SVM's in this context over other algorithms such as nearest neighbors. Our results are averaged over 3 random splits, using the Spider toolbox.

We used 9 bits for the color of each pixel to reduce the size of the RGB color space to $8^3 = 512$ from the original set of $256^3 = 16,777,216$ colors, and we defined centered grids of $4, 4^2 = 16$ and $4^3 = 64$ local patches. We provide results for each of the 5 considered kernels and for each considered depth $D$ ranging from 1 to 3. Figure 5 presents $15 = 5 \times 3$ plots, where each plot displays the misclassification rate as a function of the width parameter $\frac{1}{\lambda}$ and the branching process prior $\varepsilon$ set over all nodes of the tree. The constant C is set to 100, but other choices for C (1000 and 10) gave comparable plots, although a bit different in shape. By considering values of $\varepsilon$ ranging from 0 to 1, we aim at giving a sketch of the robustness of the averaging approach, since the SVM's seem to perform better when $0 < \varepsilon < 1$ for a large span of $\lambda$ values. For a better understanding of these plots, the reader may refer to Figure 4 which focuses on $\psi_{H_1}$ and $D = 2$, noting that the color scales used for Figures 4 and 5 are the same. Finally, the Gaussian kernel was also tested but its very poor performance (with error rate above 22% for all parameters) illustrates once more that the Gaussian kernel is usually a poor choice to compare histograms directly.

## 5 Discussion

The computation of averaged kernels can be performed almost as fast as kernels which only rely on fine resolutions, which along with their robustness and improved performance might advocate their use, notably as an extension of kernels based on arbitrary partitions (Grauman & Darrell, 2005; Matsuda et al., 2005). Principled ways of estimating in a semi-supervised setting both $\lambda$ and $\varepsilon$, or preferably localized priors $\lambda_T$ and $\varepsilon_T, T \in P_0^D$, might give them an additional edge. This is a topic of current research, and we suggest to set these parameters through cross-validation at the moment, while $H_1$ seems to be a reasonable choice to define the base kernel. Our approach is related to the Multiple Kernel Learning framework (Lanckriet et al., 2004), although we do not aim here at learning linear combinations of the kernels $k_T$, but rather start from an hierarchical belief on them to propose an algebraic combination.

**Acknowledgments**: This research was supported by the Function and Induction Research Project, Transdisciplinary Research Integration Center - Research Organization of Information and Systems.

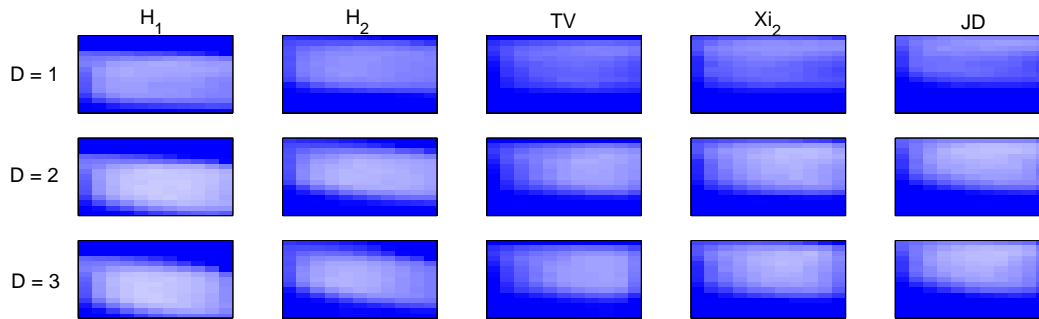

Figure 5: Error-rate results for different kernels and depths are displayed in the same way that in Figure 4, using the same colorscale across experiments.

## Footnotes

[1] $\mathcal{P}(\mathcal{L})$ is quite a big space, since if $\mathcal{L}$ is a finite set of cardinal $r$, the cardinal of the set of partitions is known as the Bell Number of order $r$ with $B_r = \frac{1}{e} \sum_{u=1}^{\infty} \frac{u^r}{u!} \underset{r \to \infty}{\sim} e^{r \ln r}$.

## References

Amari, S.-I., & Nagaoka, H. (2001). *Methods of information geometry*. AMS vol. 191.

Berg, C., Christensen, J. P. R., & Ressel, P. (1984). *Harmonic analysis on semigroups*. No. 100 in Graduate Texts in Mathematics. Springer Verlag.

Catoni, O. (2004). *Statistical learning theory and stochastic optimization*. No. 1851 in Lecture Notes in Mathematics. Springer Verlag.

Chapelle, O., Haffner, P., & Vapnik, V. (1999). SVMs for histogram based image classification. *IEEE Transactions on Neural Networks*, *10*, 1055.

Cuturi, M., Fukumizu, K., & Vert, J.-P. (2005). Semigroup kernels on measures. *JMLR*, *6*, 1169–1198.

Cuturi, M., & Vert, J.-P. (2005). The context-tree kernel for strings. *Neural Networks*, *18*, 1111 –1123.

Grauman, K., & Darrell, T. (2005). The pyramid match kernel: Discriminative classification with sets of image features. *ICCV* (pp. 1458–1465). IEEE Computer Society.

Haussler, D. (1999). *Convolution kernels on discrete structures* (Technical Report). UC Santa Cruz. CRL-99-10.

Hein, M., & Bousquet, O. (2005). Hilbertian metrics and positive definite kernels on probability measures. *Proceedings of AISTATS*.

Joachims, T. (2002). *Learning to classify text using support vector machines: Methods, theory, and algorithms*. Kluwer Academic Publishers.

Kondor, R., & Jebara, T. (2003). A kernel between sets of vectors. *Proc. of ICML'03* (pp. 361–368).

Lafferty, J., & Lebanon, G. (2005). Diffusion kernels on statistical manifolds. *JMLR*, *6*, 129–163.

Lanckriet, G., Cristianini, N., Bartlett, P., El Ghaoui, L., & Jordan, M. (2004). Learning the kernel matrix with semidefinite programming. *Journal of Machine Learning Research*, *5*, 27–72.

Leslie, C., Eskin, E., Weston, J., & Noble, W. S. (2003). Mismatch string kernels for svm protein classification. *NIPS 15*. MIT Press.

Matsuda, A., Vert, J.-P., Saigo, H., Ueda, N., Toh, H., & Akutsu, T. (2005). A novel representation of protein sequences for prediction of subcellular location using support vector machines. *Protein Sci.*, *14*, 2804–2813.

Rätsch, G., & Sonnenburg, S. (2004). *Accurate splice site prediction for caenorhabditis elegans*, 277–298. MIT Press series on Computational Molecular Biology. MIT Press.

Schölkopf, B., Tsuda, K., & Vert, J.-P. (2004). *Kernel methods in computational biology*. MIT Press.

Vert, J.-P., Saigo, H., & Akutsu, T. (2004). Local alignment kernels for protein sequences. In B. Schölkopf, K. Tsuda and J.-P. Vert (Eds.), *Kernel methods in computational biology*. MIT Press.
